# Proximity Effect Corrections in Electron Beam Lithography Using a Neural Network

**Robert C. Frye**
AT&T Bell Laboratories
600 Mountain Avenue
Murray Hill, NJ 08854

**Kevin D. Cummings***
AT&T Bell Laboratories
600 Mountain Avenue
Murray Hill, NJ 08854

**Edward A. Rietman**
AT&T Bell Laboratories
600 Mountain Avenue
Murray Hill, NJ 08854

## Abstract

We have used a neural network to compute corrections for images written by electron beams to eliminate the proximity effects caused by electron scattering. Iterative methods are effective, but require prohibitively computation time. We have instead trained a neural network to perform equivalent corrections, resulting in a significant speed-up. We have examined hardware implementations using both analog and digital electronic networks. Both had an acceptably small error of 0.5% compared to the iterative results. Additionally, we verified that the neural network correctly generalized the solution of the problem to include patterns not contained in its training set. We have experimentally verified this approach on a Cambridge Instruments EBMF 10.5 exposure system.

## 1 INTRODUCTION

Scattering imposes limitations on the minimum feature sizes that can be reliably obtained with electron beam lithography. Linewidth corrections can be used to control the dimensions of isolated features (*i.e.* intraproximity, Sewell, 1978), but meet with little success when dealing with the same features in a practical context, where they are surrounded by other features (*i.e.* interproximity). Local corrections have been proposed using a self-consistent method of computation for the desired incident dose pattern (Parikh, 1978). Such techniques require inversion of large matrices and prohibitive amounts of computation time. Lynch *et al.*, 1982, have proposed an analytical method for proximity corrections based on a solution of a set of approximate equations, resulting in a considerable improvement in speed.

The method that we present here, using a neural network, combines the computational simplicity of the method of Lynch *et al.* with the accuracy of the self-consistent methods. The first step is to determine the scattered energy profile of the electron beam which depends on the substrate structure, beam size and electron energy. This is

then used to compute spatial variations in the dosage that result when a particular image is scattered. This can be used to iteratively compute a corrected image for the input pattern. The goal of the correction is to adjust the written image so that the incident pattern of dose after scattering approximates the desired one as closely as possible. We have used this iterative method on a test image to form a training set for a neural network. The architecture of this network was chosen to incorporate the basic mathematical structure as the analytical method of Lynch *et al.*, but relies on an adaptive procedure to determine its characteristic parameters.

## 2  CALCULATING PROXIMITY CORRECTED PATTERNS

We determined the radial distribution of scattered dose from a single pixel by using a Monte-Carlo simulation for a variety of substrates and electron beam energies (Cummings, 1989). As an example problem, we looked at resist on a heavy metal substrate. (These are of interest in the fabrication of masks for x-ray lithography.) For a 20 KeV electron beam this distribution, or "proximity function," can be approximated by the analytical expression

$$f(r) = \frac{1}{\pi(1+v+\xi)} \left[ \frac{e^{-(r/\alpha)^2}}{\alpha^2} + \frac{v e^{-(r/\gamma)}}{2\gamma^2} + \frac{\xi e^{-(r/\zeta)}}{2\zeta^2} \right]$$

where

$$\alpha = 0.038\,\mu m, \quad \gamma = 0.045\,\mu m, \quad \zeta = 0.36\,\mu m, \quad v = 3.49 \quad \text{and} \quad \xi = 6.42.$$

The unscattered image is assumed to be composed of an array of pixels, $I_0(x,y)$. For a beam with a proximity function $f(r)$ like the one given above, the image after scattering will be

$$I_s(x,y) = \sum_{m=-\infty}^{\infty} \sum_{n=-\infty}^{\infty} I_0(x-m,y-n)\, f((m^2+n^2)^{1/2}),$$

which is the discrete convolution of the original image with the lineshape $f(r)$. The approach suggested by analogy with signal processing is to deconvolve the image by an inverse filtering operation. This method cannot be used, however, because it is impossible to generate negative amounts of electron exposure. Restricting the beam to positive exposures makes the problem inherently nonlinear, and we must rely instead on an iterative, rather than analytical, solution.

Figure 1 shows the pattern that we used to generate a training set for the neural network. This pattern was chosen to include examples of the kinds of features that are difficult to resolve because of proximity effects. Minimum feature sizes in the pattern ore 0.25 μm and the overall image, using 0.125 μm pixels, is 180 pixels (22.5 μm) on a side, for a total of 32,400 pixels. The initial incident dose pattern for the iterative correction of this image started with a relative exposure value of 100% for exposed pixels and 0 for unexposed ones. The scattered intensity distribution was computed from this incident dose using the discrete two-dimensional convolution with the summation truncated to a finite range, $r_0$. For the example proximity function 95% of the scattered intensity is contained within a radius of 1.125 μm (9 pixels) and this value was used for $r_0$. The scattered intensity distribution was computed and compared with the desired pattern of 100% for exposed and 0 for unexposed pixels. The

difference between the resulting scattered and desired distributions is the error. This error was subtracted from the dose pattern to be used for the next iteration. However, since negative doses are not allowed, negative regions in the correction were truncated to zero.

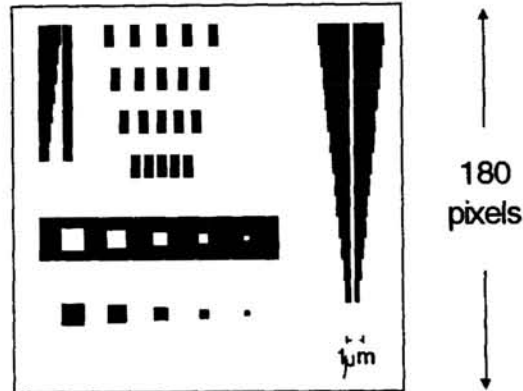

Figure 1: Training pattern

Using this algorithm, a pixel that receives a dosage that is too small will have a negative error, and on the next iteration its intensity will be increased. Unexposed pixels (*i.e.* regions where resist is to be removed) will always have some dosage scattered into them from adjacent features, and will consequently always show a positive error. Because the written dose in these regions is always zero, rather than negative, it is impossible for the iterative solution to completely eliminate the error in the final scattered distribution. However, the nonlinear exposure properties of the resist will compensate for this. Moreover, since all exposed features receive a uniform dose after correction, it is possible to choose a resist with the optimal contrast properties for the pattern.

Although this iterative method is effective, it is also time consuming. Each iteration on the test pattern required about 1 hour to run on a 386 based computer. Four iterations were required before the smallest features in the resist were properly resolved. Even the expected order of magnitude speed increase from a large mainframe computer is not sufficient to correct the image from a full sized chip consisting of several billion pixels. The purpose of the neural network is to do these same calculations, but in a much shorter time.

## 3  NETWORK ARCHITECTURE AND TRAINING

Figure 2 shows the relationship between the image being corrected and the neural network. The correction for one pixel takes into account the image surrounding it. Since the neighborhood must include all of the pixels that contribute appreciable scattered intensity to the central pixel being corrected, the size of the network was determined by the same maximum radius, $r_0 = 1.125\,\mu m$, that characterized the scattering proximity function. The large number of inputs would be difficult to manage in an analog network if these inputs were general analog signals, but fortunately the input data are binary, and can be loaded into an analog network using digital shift registers.

Figure 3 shows a schematic diagram of the analog network. The binary signals from the shift registers representing a portion of the image were connected to the buffer amplifiers through 10 KΩ resistors. Each was connected to only one summing node, corresponding to its radial distance from the center pixel. This stage converted the 19 x 19 binary representation of the image into 10 analog voltages that represented the radial distribution of the surrounding intensity. The summing amplifier at the output was connected to these 10 nodes by variable resistors. This resulted in an output that was a weighted sum of the radial components.

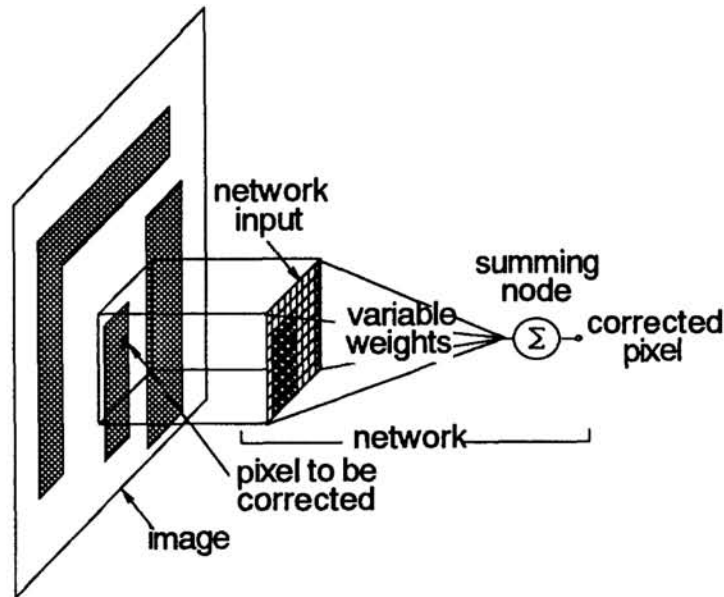

Figure 2:  Network configuration

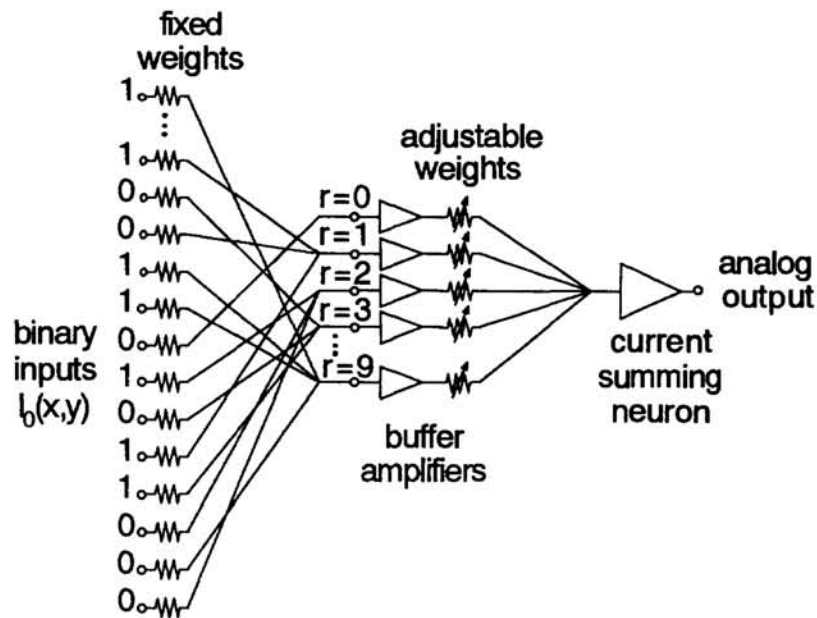

Figure 3:  Schematic diagram of the analog network

Functionally, this network does the operation

$$V_{out} = \sum_{r=0}^{9} w_r \langle I_0 \rangle_r,$$

where $w_r$ are the weight coefficients set by the adjustable resistors and $\langle I_0 \rangle_r$ are the average values of the pixel intensity at radius r. The form of this relationship is identical to the one proposed by Lynch *et al.* but uses an adaptive method, rather than an analytical one, to determine the coefficients $w_r$.

The prototype analog hardware network was built on a wire wrap board using 74HC164 8 bit CMOS static shift registers and LM324 quad operational amplifiers for the active devices. The resistors in the first layer were 10 K$\Omega$ thin-film resistors in dual-in-line packages and had a tolerance of 1%. The ten adjustable resistors in the second layer of the network were 10 turn precision trimmers. Negative weights were made by inverting the sign of the voltage at the buffer amplifiers. For comparison, we also evaluated a digital hardware implementation of this network. It was implemented on a floating point array processor built by Eighteen Eight Laboratories using an AT&T DSP-32 chip operating at 8 MFLOPs peak rate. The mathematical operation performed by the network is equivalent to a two-dimensional convolution of the input image with an adaptively learned floating point kernel.

The adjustable weight values for both networks were determined using the delta rule of Widrow and Hoff (1960). For each pixel in the trial pattern of Figure 1 there was a corresponding desired output computed by the iterative method. Each pixel in the test image, its surroundings and corresponding analog corrected value (computed by the iterative method) constituted a single learning trial, and the overall image contained 32,400 of them. We found that the weight values stabilized after two passes through the test image.

## 4 NEURAL NETWORK PERFORMANCE

The accuracy of both the analog and digital networks, compared to the iterative solution, was comparable. Both showed an average error for the test image of 0.5%, and a maximum error of 9% on any particular pixel. The accuracy of the networks on images other than the one used to train them was comparable, averaging about 0.5% overall.

Convolution with an adaptively-learned kernel is itself a relatively efficient computational algorithm. The iterative method required 4 hours to compute the correction for the 32,400 pixel example. Equivalent results were obtained by convolution in about 6.5 minutes using the same computer. Examination of the assembled code for the software network showed that the correction for each pixel required the execution of about 30 times fewer instructions than for the iterative method.

The analog hardware generated corrections for the same example in 13.5 seconds. Almost 95% of this time was used for input/output operations between the network and the computer. It was the time required for the I/O, rather than the speed of the circuit, that limited the dynamic performance of this system. Clearly, with improved I/O hardware, the analog network could be made to compute these corrections much more quickly.

The same algorithm, running on the digital floating point array processor performed the correction for this example problem in 4.5 seconds. The factor of three improvement over the analog hardware was primarily a result of the decreased time needed for I/O in the DSP-based network. The digital network was not appreciably more accurate than the analog one, indicating that the overall accuracy of operation was determined primarily by the network architecture rather than by limitations in the implementation. These results are summarized in Table 1.

**Table 1:** Comparison of computational speed for various methods.

| METHOD | SPEED |
|---|---|
| Iteration | 6 years /mm$^2$ |
| Software network | 100 days /mm$^2$ |
| Analog hardware network | 2 days /mm$^2$ |
| Digital hardware network | 18 hours /mm$^2$ |

## 5  EXPERIMENTAL VERIFICATION

Recently, we have evaluated this method experimentally using a Cambridge Instruments EBMF 10.5 exposure system (Cummings, *et al.*, 1990). The test image was 1 mm$^2$ and contained 11,165 Cambridge shapes and $6.7 \times 10^7$ pixels. The substrate was silicon with 0.5 μm of SAL601-ER7 resist exposed at 20 KeV beam energy. The range of the scattered electrons is more than three times greater for these conditions than in the tests described above, requiring a network about ten times larger. The neural network computations were done using the digital floating point array processor, and required about 18 hours to correct the entire image. Input to the program was Cambridge source code, which was converted to a bit-mapped array, corrected by the neural network and then decomposed into new Cambridge source code.

Figure 4 shows SEM micrographs comparing one of the test structures written with and without the neural network correction. This test structure consists of a 10 μm square pad next to a 1 μm wide line, separated by a gap of 0.5 μm. Note in the uncorrected pattern that the line widens in the region adjacent to the large pad, and the webs of resist extending into the gap. This is caused by excess dosage scattered into these regions from the large pad. In the corrected pattern, the dosage in these regions has been adjusted, resulting in a uniform exposure after scattering and greatly improved pattern resolution.

## 6  CONCLUSIONS

The results of our trial experiments clearly demonstrate the computational benefits of a neural network for this particular application. The trained analog hardware network performed the corrections more than 1000 times faster than the iterative method using the same computer, and the digital processor was 3000 times faster. This technique is readily applicable to a variety of direct write exposure systems that have the capability to write with variable exposure times. Implementation of the network on more sophisticated computers with readily available coprocessors can directly lead to another order of magnitude improvement in speed, making it practical to correct full chip-sized images.

The performance of the analog network suggests that with improved speed of I/O between the computer and the network, it would be possible to obtain much faster operation. The added flexibility and generality of the digital approach, however, is a considerable advantage.

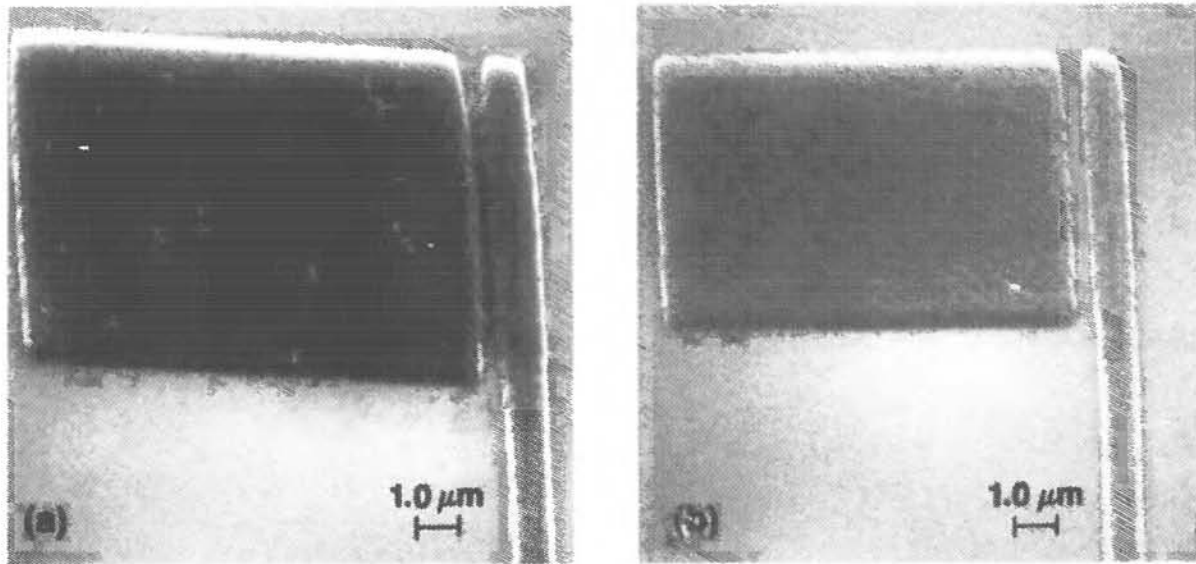

Figure 4:  Comparison of a test structure written with and without correction

## Acknowledgments

We thank S. Waaben and W. T. Lynch for useful discussions, suggestions and information, and J. Brereton who assisted in building the hardware and trial patterns for initial evaluation.  We also thank C. Biddick, C. Lockstampfor, S. Moccio and B. Vogel for technical support in the experimental verification.

## Footnotes

* Present address: Motorola Inc. Phoenix Corporate Research Laboratories, 2100 East Elliot Rd. Tempe, AZ 85284.

## References

H. Sewell, "Control of Pattern Dimensions in Electron Lithography," J. Vac. Sci. Technol. **15**, 927 (1978).

M. Parikh, "Self-Consistent Proximity Effect Correction Technique for Resist Exposure (SPECTRE)," J. Vac. Sci. Technol. **15**, 931 (1978).

W.T. Lynch, T. E. Smith and W. Fichtner, "An Algorithm for Proximity Effect Correction with E-Beam Exposure," Int'l. Conf. on Microlithography, Microcircuit Engineering  pp 309-314, Grenoble (1982).

K. D. Cummings "Determination of Proximity Parameters for Electron Beam Lithography," AT&T Bell Laboratories Internal Memorandum.

B. Widrow and M. E. Hoff, "Adaptive Switching Circuits," IRE WESCON Convention Record, Part 4, 96-104 (1960).

K. D. Cummings, R. C. Frye and E. A. Rietman, "Using a Neural Network to Proximity Correct Patterns Written with a Cambridge EBMF 10.5 Electron Beam Exposure System," Applied Phys. Lett. **57** 1431 (1990).